# Using "epitomes" to model genetic diversity: Rational design of HIV vaccine cocktails

**Nebojsa Jojic, Vladimir Jojic, Brendan Frey, Chris Meek and David Heckerman**
Microsoft Research

## Abstract

We introduce a new model of genetic diversity which summarizes a large input dataset into an epitome, a short sequence or a small set of short sequences of probability distributions capturing many overlapping subsequences from the dataset. The epitome as a representation has already been used in modeling real-valued signals, such as images and audio. The discrete sequence model we introduce in this paper targets applications in genetics, from multiple alignment to recombination and mutation inference. In our experiments, we concentrate on modeling the diversity of HIV where the epitome emerges as a natural model for producing relatively small vaccines covering a large number of immune system targets known as epitopes. Our experiments show that the epitome includes more epitopes than other vaccine designs of similar length, including cocktails of consensus strains, phylogenetic tree centers, and observed strains. We also discuss epitome designs that take into account uncertainty about T-cell cross reactivity and epitope presentation. In our experiments, we find that vaccine optimization is fairly robust to these uncertainties.

## 1 Introduction

Within and across instances of a certain class of a natural signal, such as a facial image, a bird song recording, or a certain type of a gene, we find many repeating fragments. The repeating fragments can vary slightly and can have arbitrary (and usually unknown) sizes. For instance, in cropped images of human faces, a small patch capturing an eye appears in an image twice (with a symmetry transformation applied), and across different facial images many times, as humans have a limited number of eye types. Another repeating structure across facial images is the nose, which occupies a larger patch. In mammalian DNA sequences, we find repeating regulatory elements within a single sequence, and repeating larger structures (genes, or gene fragments) across species. Instead of defining size, variability and typical relative locations of repeating fragments manually, in an application-driven way, the 'epitomic analysis' [5] is an unsupervised approach to estimating repeating fragment models, and simultaneously aligning the data to them. This is achieved by considering data in terms of randomly selected overlapping fragments, or patches, of various sizes and mapping them onto an 'epitome,' a learned structure which is considerably larger than any of the fragments, and yet much smaller than the total size of the dataset.

We first introduced this model for image analysis [5], and it has since been used for video and audio analysis [2, 6], as well. This paper introduces a new form of the epitome as a sequence of multinomial distributions (Fig. 1), and describe its applications to HIV diversity modeling and rational vaccine design. We show that the vaccines optimized using our algorithms are likely to have broader predicted coverage of immune targets in HIV than the previous rational designs.

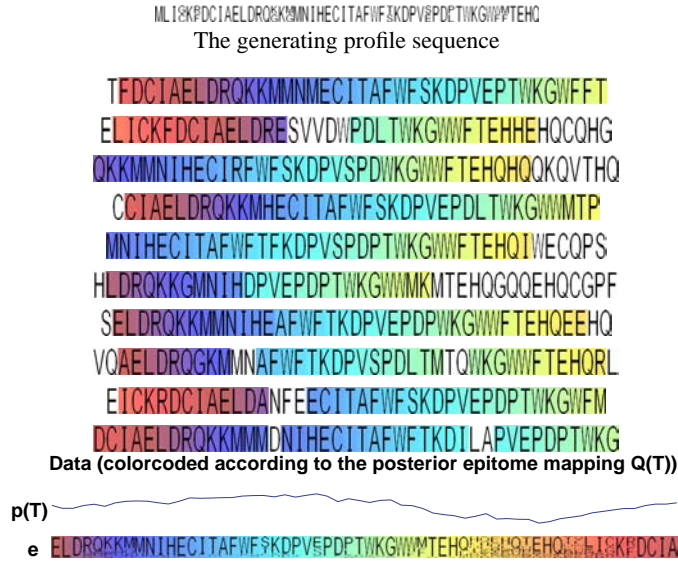

Figure 1: The epitome (e) learned from data synthesized from the generating profile sequence (Section 5). A color coding in the epitome and data sequences is used to show the mapping between epitome and data positions. A white color indicates that the letter was likely generated from the garbage component of the epitome. The distribution $p(\mathcal{T})$ shows which 9mers from the epitome were more likely to generate patches of the data.

## 2    Sequence epitome

The central part of Fig. 1 illustrates a small set of amino acid sequences $X = \{x_{ij}\}$ of size MN (with $i$ indexing a sequence, and $j$ indexing a letter within a sequence, and $M = \max i$, $N = \max j$). The sequences share patterns (although sometimes with discrepancies in isolated amino-acids) but one sequence may be similar to other sequences in different regions. The sequences are generated synthetically by combining the pieces of the profile sequence given in the first line of the figure, with occasional insertions of random sequence fragments, as discussed in Section 5. Sequence variability in this synthetic example is slightly higher than that found in the NEF protein of the human immunodeficiency virus (HIV) [7], while the envelope proteins of the same virus exhibit more variability. Examples of high genetic diversity can also be found in higher-level organisms, for example in the regions coding for immune system's pattern recognition molecules.

The last row in the figure illustrates an epitome optimized to represent the variability in the sequences above. In general, the epitome is a smaller array $E = \{e_{mn}\}$ of size $M_e \times N_e$, where $M_e N_e \ll MN$. In the figure, $M_e = 1$. An epitome can be parameterized in different ways, but in the figure, each epitome element $e_{mn}$ is a multinomial distribution with the probability of each letter represented by its height. The epitome's summarization quality is defined by a simple generative model which considers the data $X$ in terms of shorter subsequences, $X_{\mathcal{S}}$. A subsequence $X_{\mathcal{S}}$ is defined as an ordered subset of letters from $X$ taken from positions listed in the ordered index set $\mathcal{S}$. For instance, the set $\mathcal{S} = \{(4,8), (4,9), (4,10), (4,11)\}$ points to a contiguous patch of letters in the fourth sequence $X_{\mathcal{S}} = RQKK$. Similarly, set $\mathcal{S} = \{(6,2), (6,3), (6,4), (6,5), (6,6)\}$ points to the patch $X_{\mathcal{S}} = LDRQK$ in the sixth sequence. A number of such patches[1] of various lengths can be taken randomly (and with overlap). The quality of the epitome is then defined as the total likelihood of these patches under the generative model which generates each patch from a set of distributions $E_{\mathcal{T}}$, where $\mathcal{T}$ is an ordered set of indices into the epitome (In the figure, the epitome is defined on a circle, so that the index progression continues from $N_e$ to 1. (This reduces local minima problems in the EM algorithm for epitome learning as discussed in Sections 4 and 5). For each data patch, the mapping $\mathcal{T}$ is considered a hidden variable,

and the generative process is assumed to consist of the following two steps

- Sample a patch $E_{\mathcal{T}}$ from $E$ according to $p(\mathcal{T})$. To illustrate $p(\mathcal{T})$ in Fig. 1, we consider only the set of of all 9-long contiguous patches. For such patches, which are sometimes called nine-mers, we can index different sets $\mathcal{T}$ by their first elements and plot $p(\mathcal{T})$ as a curve with the domain $\{1, ..., N_e - 8\}$.

- Generate a patch $X_{\mathcal{S}}$ from $E_{\mathcal{T}}$ according to $p(X_{\mathcal{S}}|E_{\mathcal{T}}) = \prod_{k=1}^{|\mathcal{T}|} e_{\mathcal{T}(k)}(X_{\mathcal{S}(k)})$, with $\mathcal{T}(k)$ and $\mathcal{S}(k)$ denoting the $k$-th element in the epitome and data patches.

Each execution of these two steps can, in principle, generate any pattern. The probability (likelihood) of generating a particular pattern indicated by $\mathcal{S}$ is

$$p(X_S) = \sum_{\mathcal{T}} p(X_{\mathcal{S}}|E_{\mathcal{T}})p(\mathcal{T}). \tag{1}$$

Given the epitome, we can perform inference in this model and compute the posterior distribution over mappings $\mathcal{T}$ for a particular model. For instance, for $X_{\mathcal{S}} = RQKK$, the most probable mapping is $\mathcal{T} = \{(1,4),(1,5),(1,6),(1,7)\}$. In Section 4, we discuss algorithms for estimating the epitome distributions.

Our illustration points to possible applications of epitomes to multiple sequence alignment, and therefore requires a short discussion on similarity to other biological sequence models [3]. While the epitome is a fully probabilistic model and thus defines a precise cost function for optimization, as was the case with HMM-based models, or dynamic programming solutions to sequence alignment, the main novelty in our approach is the consideration of both the data and the model parameters in terms of overlapping patches. This leads to the alignment of different parts of the sequences to the joint representation without explicit constraints on contiguity of the mappings or temporal models used in HMMs. Also, as we discuss in the next section, our goal is diversity modeling, and *not* multiple alignment. The epitome's robustness to the length, position and variability of repeating sequence fragments allows us to bypass both the task of optimal global alignment, and the problem of *defining* the notion of global alignment. In addition, consideration of overlapping patches in a biological sequence can be viewed as modeling independent binding processes, making the patch independence assumption of our generative model biologically relevant. We illustrate these properties of the epitome on the problem of HIV diversity modeling and rational vaccine design.

## 3  HIV evolution and rational vaccine design

Recent work on the rational design of HIV vaccines has turned to cocktail approaches with the intention of protecting a person against many possible variants of the HIV virus. One of the potential difficulties with cocktail design is vaccine size. Vaccines with a large number of nucleotides or amino acids are expensive to manufacture and more difficult to deliver. In this section, we will show that epitome modeling can overcome this limitation by providing a means for generating smaller vaccines representing a wide diversity of HIV in an immunologically relevant way. We focus on the problem of constructing an optimal cellular vaccine in terms of its coverage of MHC-I epitopes, short contiguous patterns of 8-11 aminoacids in HIV proteins [8].

Major histocompatibility complex (MHC) molecules are responsible for presentation of short segments of internal proteins, called "epitopes," on the surface of a cell. These peptides (protein segments) can then be observed from outside the cell by killer T-cells, which normally react only to foreign peptides, instructing the cell to self-distruct. The killer cells and their offspring have the opportunity to bind to multiple infected cells, and so their first binding to a particular foreign epitope is used to accelerate an immune reaction to other infected cells exposing the same epitope. Such responses are called memory responses and can persist for a long time after the infection has been cleared, providing longer-term immunity to the disease. The goal of vaccine design is to create artifical means to produce such immunological memory of a particular virus without the danger of developing the disease.

In the case of a less variable virus, the vaccination may be possible by delivering a foreign protein similar to the viral protein into a patient's cells, triggering the immune response. However, HIV is capable of assuming many different forms, and immunization against a single strain is largely expected to be insufficient. In fact, without appropriate optimization, the number of different proteins needed to cover the viral diversity would be too large for the known vaccine delivery mechanisms. It is well known that epitopes within and across the strains in a population overlap [7]. The epitome model naturally exploits this overlap to construct a vaccine that can prime the immune system to attack as many potential epitopes as possible. For instance, if the sequences in Fig 1 were HIV fragments from different strains of the virus, then the epitome would contain many potential epitopes of lengths 8-11 from these sequences. Furthermore, the context of the captured epitopes in the epitome is similar to the context in the epitomized sequences, which increases the chances of equivalent presentation of the epitome and data epitopes.

MHC molecules are encoded within the most diverse region of the human genome. This gives our species a diversity advantage in numerous clashes with viruses. Each individual has a slightly different set of MHC molecules which bind to different motifs in the proteins expressed and cleaved in the cell. Due to the limitation in MHC binding, each person's cells are capable of presenting only a small number of epitopes from the invading virus, but an entire human population attacks a diverse set of epitopes. The MHC molecule selects the protein fragments for presentation through a binding process which is loosely motif-specific. There are several other processes that precede or follow the MHC binding, and the combination of all of these processes can be characterized either by the concentration of presented epitopes, or by the combination of the binding energies involved in these processes[2]. Some of these processes can be influenced by a context of the epitope (short amino acid fragments in the regions on either side of the epitope).

Another issue to be considered in HIV evolution and vaccine design is the T-cell cross reactivity: The killer cells primed with one epitope may be capable of binding to other related epitopes, and therefore a small set of priming epitopes may induce a broader immunity. As in the case of MHC binding, the likelihood of priming a T-cell, as well as cross-reaction with a different epitope, can be linked to the binding energies.

The epitome model maps directly to these immunity variables. If the epitome content is to be delivered to a cell in the vaccination phase, then each patch $E_{\mathcal{T}}$ indexed by data index set $\mathcal{T}$ corresponds either to an epitope or to a longer contiguous patch (e.g. 12 amino acids or more) containing both an epitope and its context that influences presentation. The prior $p(\mathcal{T})$ reflects the probability of presentation of the epitome fragments, and should reflect processes invloved in presentation, including MHC binding. The presented epitome fragments $E_{\mathcal{T}}$ in different patients' cells may prime T-cells capable of cross-reacting with some of the epitopes $\mathbf{X}_{\mathcal{S}}$ presented by the infected cells infected by one of the known strains in the dataset $\mathbf{X}$. The cross-reaction distribution corresponds to the epitome distribution $p(\mathbf{X}_{\mathcal{S}}|E_{\mathcal{T}})$. Vaccination is successful if the vaccine primes the immune system to attack targets found in the known circulating strains. A natural criterion to optimize is the similarity between the distribution over the epitopes learned by the immune systems of patients vaccinated with the epitome (taking into account the cross-reactivity) and the distribution over the epitopes from circulating strains. Therefore, the vaccine quality directly depends on the likelihood of the designated epitopes $p(X_{\mathcal{S}})$ under the epitome. To see this, consider directly optimizing the KL divergence between the distribution $p_d(\mathbf{X}_s)$ over epitopes found in the data and the distribution over the targets for which the T-cells are primed according to $p(\mathbf{X}_s)$. This KL distance differs from the log likelihood of all the data patches weighted by $p_d(\mathbf{X}_s)$,

$$\log p(\{X_{\mathcal{S}}\}_d) = \sum_{\mathcal{S}} p_d(X_{\mathcal{S}}) \log \sum_{\mathcal{T}} p(X_{\mathcal{S}}|E_{\mathcal{T}})p(\mathcal{T}), \qquad (2)$$

only by a constant (the entropy of $p_d(\mathbf{X}_s)$). The distribution $p_d(\mathbf{X}_s)$ can serve as the indicator of epitopes and be equal to either zero or a constant for all patches, and then the above weighted likelihood is equivalent to the total likelihood of selected patches. This

___________________________

[2]The probabilities of physical events are often modeled as having an exponential relationship with the energy changes.

distribution can also reflect the probability of presentation of epitopes $X_{\mathcal{S}}$, or the uncertainty of the experiment or the prediction algorithm used to predict which parts of the circulating strains correspond to MHC epitopes.

While the epitome can serve as a diversity model and be used to construct evolutionary models and peptides for experimental epitope discovery, it can also serve as as an actual immungen (the pattern containing the immunologically important message to the cell) in vaccine. The most general version of epitome as a sequence of mutlinomial distributions could be relevant for sequence classification, recombination modeling, and design of peptides for binding essays. In some of these applications, the distribution $p(X_{\mathcal{S}}|E_{\mathcal{T}})$ may have a semantics different than cross-reactivity, and could for instance represent mutations dependent on the immune type of the host, or the subtype of the virus. On the other hand, when the epitome is used for immunogen design, then cross-reactivity $p(X_{\mathcal{S}}|E_{\mathcal{T}})$ can be conveniently captured by constraining each distribution $e_{mn}$ to have probabilities for the twenty aminoacids from the set $\{\frac{\epsilon}{19}, 1 - \epsilon\}$. The mode of the epitome can then be used as a deterministic vaccine immunogen[3], and the probability of cross-reaction will then directly depend on the number of letters in $\mathbf{X}_{\mathcal{S}}$ that are different from the mode of $E_{\mathcal{T}}$.

While the epitome model components are mapped here to the elements of the interaction between HIV and the immune system of the host, other applications in biology would probably be based on a different semantics for the epitome components. We would expect that the epitome would map to biological sequence analysis problems more naturally than to image and audio modeling tasks, where the issue of the partition function arises. Epitome as a generative model over-generates - generated patches overlap, and so each data element is generated multiple times. In the image applications, we have avoided this problem through constraints on the posterior distributions, while the traditional approach would be to deal with the partition function (perhaps through sampling). However, the strains of a virus are observed by the immune system through overlapping patches, independently sampled from the viral proteins by biological processes. This fits epitome as a vaccination model. More generally, epitome is compatible with the evolutionary forces that act independently on overlapping patches of a biological sequence.

## 4   Epitome learning

Since epitomes can have multiple applications, we provide a general discussion of optimization of all parameters of the epitome, although in some applications, some of the parameters may be known a priori. As a unified optimization criterion we use the free energy [9] of the model (2),

$$F(\{X_{\mathcal{S}}\}_d|E) = \sum_S p_d(X_{\mathcal{S}}) \sum_{\mathcal{T}} q(\mathcal{T}|\mathcal{S}) \log \frac{q(\mathcal{T}|\mathcal{S})}{p(X_{\mathcal{S}}|E_{\mathcal{T}})\, p(\mathcal{T})}, \qquad (3)$$

where $q(\mathcal{T}|\mathcal{S})$ is an variational distribution, where

$$-\log p(\{X_{\mathcal{S}}\}_d|E) = \arg\min_q F(\{X_{\mathcal{S}}\}_d|E). \qquad (4)$$

The model can be learned by iteratively reducing $F$, varying in each iteration either $q$ or the model parameters. When modeling biological sequences, the free energy *may* be associated with real physical events, such as molecular binding processes, where log probabilities correspond to molecular binding energies.

Setting to zero the derivatives of $F$ with respect to the $q$ distributions, the distribution $p(\mathcal{T})$, and the distributions $e_m(\ell)$ for all positions $m$, we obtain the EM algorithm [5]:

- For each $X_{\mathcal{S}}$, compute the posterior distribution over patches $q(\mathcal{T}|\mathcal{S})$:

$$q(\mathcal{T}|\mathcal{S}) \leftarrow \frac{p(X_{\mathcal{S}}|E_{\mathcal{T}})\, p(\mathcal{T})}{\sum_{\mathcal{T}} p(X_{\mathcal{S}}|E_{\mathcal{T}})\, p(\mathcal{T})}. \qquad (5)$$

- Using these $q$ distributions, update the profile sequence:

$$e_m(\ell) \leftarrow \frac{\sum_{\mathcal{S}} p_d(X_{\mathcal{S}}) \sum_k \sum_{\mathcal{T}|\mathcal{T}(k)=m} q(\mathcal{T}|\mathcal{S})[X_{\mathcal{S}(k)} = \ell]}{\sum_{\mathcal{S}} p_d(X_{\mathcal{S}}) \sum_k \sum_{\mathcal{T}|\mathcal{T}(k)=m} q(\mathcal{T}|\mathcal{S})}, \tag{6}$$

where $[\cdot]$ is the indicator function ($[true] = 1$; $[false] = 0$). If desired, also update $p(\mathcal{T})$:

$$p(\mathcal{T}) \leftarrow \frac{\sum_{\mathcal{S}} p_d(X_{\mathcal{S}}) \, q(\mathcal{T}|\mathcal{S})}{\sum_{\mathcal{S}} p_d(X_{\mathcal{S}})}. \tag{7}$$

The E step assigns a responsibility for $\mathcal{S}$ to each possible epitome patch. The M step re-estimates the epitome multinomials using these responsibilities. As mentioned, this step can re-estimate the usage probabilities of patches in the epitome, or this distribution can be kept constant. It is often useful to construct the index sets $\mathcal{T}$ such that they wrap around from one end to another. Such circular topologies can deter the EM algorithm from settling in a poor local maximum of log likelihood. It is also sometimes useful to include a garbage component (a component that generates patches containing random letters) in the model.

In general, the EM algorithm is prone to problems of local maxima. For example, if we allowed the epitome to be longer, then some of the sites with two equally likely letters could be split into two separate regions of the epitome (and in some applications, such as vaccine optimization, this is preferred, as the epitomes need to become deterministic). Epitomes situated at different local maxima, however, often define similar probability distributions $p(\{X_{\mathcal{S}}\}|E)$, and can be used for various inference tasks such as sequence recognition/classification, noise removal, and context-dependent mutation prediction.

Of course, there are optimization algorithms other than EM that can learn a profile sequence by minimizing the free energy, $E = \arg\min_E \min_q F(\{X_{\mathcal{S}}\}_d|E)$. In some situations, such as vaccine design, it is desirable to produce deterministic epitomes (containing point-mass probability distributions). Such profile sequences can be obtained by annealing the parameter $\epsilon$ that controls the amount of probability allowed to be distributed to the letter different from the most likely letter $\hat{\ell}_m = \arg\max_\ell e_m(\ell)$:

$$E = \lim_{\epsilon \to 0} \arg\min_E \min_q F(\{X_{\mathcal{S}}\}_d|E). \tag{8}$$

Finally, in cases when the probability mass is uniformly spread over the letters other than the modes of the epitome distributions, i.e., $e_{mn}(\ell) \in \{\frac{\epsilon}{19}, 1 - \epsilon\}$, the myopic optimization is a faster way of creating epitomes of high fragment (epitope) coverage than the EM with multiple initializations. The myopic optimization consists of iteratively increasing the length of the epitome by appending a patch (possibly with overlap) from the data which maximally reduces the free energy. The process stops once the desired length is achieved (rather than when the entire set of patches is included as in the superstring problem).

## 5 Experiments

To illustrate the EM algorithm for epitome learning, we created the synthetic data shown (in part) in Figure 1. The data, eighty sequences in all, were synthesized from the generating profile sequence of length fifty shown on the top line of the figure. In particular, each data sequence was created by extracting one to four (mean two) patches from the generating sequence of length three to thirty (mean sixteen), sampling from these patches to produce corresponding patches of amino acids in the data sequence, and then filling in the gaps in the data sequence with amino acids sampled from a uniform distribution over amino acids. In addition, five percent of the sites in the each data sequence were subsequently replaced with an amino acid sampled from a uniform distribution. The resulting data sequences ranged in length from 38 to 43; and on average 80% of aminoacids in each sequence come from the generating sequence. Thus, the synthesized data roughly simulates genetic diversity resulting from a combination of mutation, insertion, deletion, and recombination.

We learned an epitome model using the EM algorithm applied to all 9mer patches from the data, equally weighted. We used a two-component epitome mixture, where the first component is an (initially unknown) sequence of probability distributions, and the second component is a garbage component, useful for representing the random insertions and

mutations. Each site in the first component was initialized to a distribution slightly (and randomly) perturbed from uniform. The length of this component was set to be slightly longer than the original generating sequence. In previous experiments, we have found that a longer length helps to prevent the EM algorithm from settling in a poor local maximum of log likelihood, and it is subsequently possible to cut out unnecessary parts which can be detected in the learned prior $p(\mathcal{T})$. Also, we used an epitome with a circular topology. The first (non-garbage) component of the epitome learned after sixty iterations, shown in Figure 1, closely resembels the generating sequence even though it never saw this generating sequence during learning. (Roughly, the generating sequence starts near the end of the epitome with the patch "LIC" coded in red, and wraps around to the patch "EHQ" coded in yellow. The portion of the epitome between yellow and red is not responsible for many patches, as reflected in the distribution $p(\mathcal{T})$.) The sixty iterations of EM are illustrated in the video available at www.research.microsoft.com/~jojic/pEpitome.mpg. For each iteration, we show the first (non-garbage) component of the epitome $E$, the distribution $p(\mathcal{T})$, and the first ten sequences in the dataset, color-coded according to the mode of $q(\mathcal{T}|\mathcal{S})$, as in Figure 1. The video illustrates how the EM algorithm simultaneously learns the epitome model and aligns the data sequences.

When used for vaccine optimization, some epitome parameters can be preset based on biological knowledge. In particular, in the experiments we report on 176 gag HIV proteins from the WA cohort [8], we assume no cross reactivity (i.e., we set $\epsilon = 0$) and we consider two different possibilities for the patch data distribution $p_d(\mathbf{X}_\mathcal{S})$. The first parameter setting we consider is that $p_d(\mathbf{X}_\mathcal{S})$ is uniform for all ten amino-acid blocks found in the sequence data. The advantage of the uniform data distribution is that we only need sequence data for vaccine optimization, and not the epitope identities. The free energy criterion can be easily shown to be proportional (with a negative constant) to the coverage - the percentage of all 10mers from the data covered by the epitome, where the 10mer is considered covered if it can be found as a contiguous patch in the epitome's mode. Another advantage of this approach is that it can not miss epitopes due to errors in prediction algorithms or experimental epitope discovery, as long as sufficient coverage can be guaranteed for the given vaccine length.

The second setting of the parameters $p_d(\mathbf{X}_\mathcal{S})$ we consider is based the SYFPEITHI database [10] of known epitopes. We trained $p_d(\mathbf{X}_\mathcal{S})$ on this data using a decision tree model to represent the probability that an observed 10mer contains a presentable epitope. The advantage of this approach is that we can potentially focus our modeling power only to immunologically important variability, as long as the known epitope dataset is sufficient to capture properly the epitope distribution for at least the most frequent MHC-I molecules. Thus, for a given epitome length, we may obtain more potent vaccines than using the first parameter setting. Since $\epsilon = 0$, the resulting optimization reduces to optimizing *expected* epitope coverage, i.e., the sum of all probabilities

For both epitome settings, we epitomized the 176 gag proteins in the dataset, using the myopic algorithm, and compared the expected epitope coverage of our vaccine candidates with those of other designs, including cocktails of tree centers, consensus, and actual strains (Fig. 2). Phylogenies were constructed using neighbor joining, as is used in Phylip [4]. Clusters were generated using a mixture model of independent multinomials [1]. Observed sequences in the sequence cocktails were chosen at random. Both epitome models yield better coverage and the expected epitope coverage than other designs for any fixed length. Results are similar for the pol, nef, and env proteins. An interesting finding to note is that the epitome optimized for coverage (using uniform distribution $p_d(\mathbf{X}_\mathcal{S})$) provides essentially equally good expected coverage as the epitome directly optimized for the expected coverage. This is less surprising than it may seem - both true and predicted epitopes overlap in the sequence data, and so epitomizing all 10mers leads to similar epitomes as optimizing for coverage of the select few, but frequently overlapping epitopes. This is a direct consequence of the epitome representation, which was found appealing in previous applications for the same robustness to the number and sizes of the overlapping patches. It also indicates the possibility that an effective vaccine can be optimized without precise knowledge of all HIV epitopes.

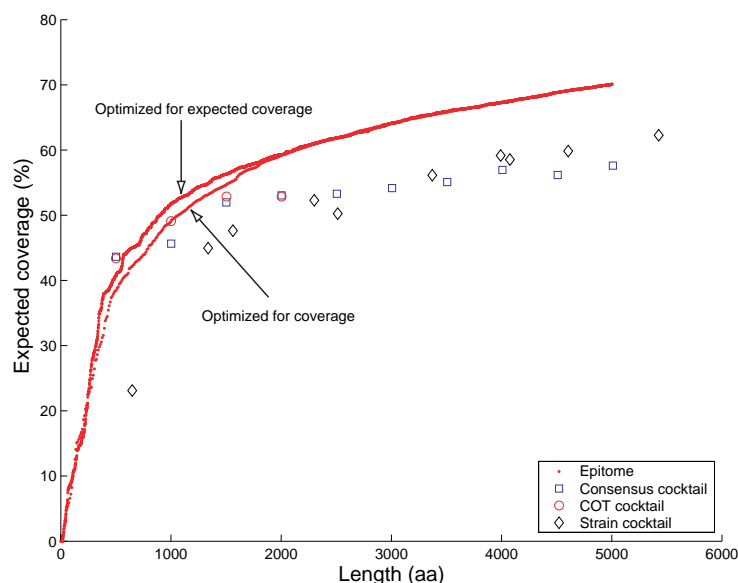

Figure 2: Expected coverage for 176 Perth gag proteins using candidate sequences of length ten. For comparison, we show expected coverage for the epitome optimized to cover all 10mers.

## 6 Conclusions

We have introduced the epitome as a new model of genetic diversity, especially well suited to highly variable biological sequences. We show that our model can be used to optimize HIV vaccines with larger predicted coverage of MHC-I epitopes than other constructs of similar lengths and so epitome can be used to create vaccines that cover a large fraction of HIV diversity. We also show that epitome optimization leads to good vaccines even when all subsequence of length 10 are considered epitopes. This suggests that the vaccines could be optimized directly from sequence data, which are technologically much easier to obtain than epitope data. Our analysis of cross-reactivity which provided similar empirical evidence of epitome robustness to cross-reactivity assumptions (see www.research.microsoft.com/~jojic/HIVepitome.html for the full set of results).

## Footnotes

[1] In principal, noncontiguous patches can be taken as well, if the application so requires.

[3]To our knowledge, there is no effective way of delivering epitome as a distribution over proteins or fragments into the cell

## References and Notes

[1]  P. Cheeseman and J. Stutz. Bayesian classification (AutoClass): Theory and results. In *Advances in Knowledge Discovery and Data Mining,* Fayyad, U., Piatesky-Shapiro, G., Smyth, P., and Uthurusamy, R., eds. (AAAI Press, 1995).

[2]  V. Cheung, B. Frey, and N. Jojic. Video epitome. *CVPR* 2005.

[3]  R. Durbin et al. *Biological Sequence Analysis : Probabilistic Models of Proteins and Nucleic Acids.* Cambridge University Press, 1998.

[4]  J. Felsenstein. Phylip (phylogeny inference package) version 3.6, 2004.

[5]  N. Jojic, B. Frey, and A. Kannan. Epitomic analysis of appearance and shape. In *Proceedings of the Ninth International Conference on Computer Vision,* Nice (2003). Video available at http://www.robots.ox.ac.uk/ awf/iccv03videos/.

[6]  A. Kapoor and S. Basu. The audio epitome: A new representation for modeling and classifying auditory phenomena. *ICASSP* 2004.

[7]  B.T.M. Korber, C. Brander, B.F. Haynes, R. Koup, C. Kuiken, J.P. Moore, B.D. Walker, and D.I. Watkins. *HIV Molecular Immunology.* Los Alamos National Laboratory, Theoretical Biology and Biophysics, Los Alamos, NM, 2002.

[8]  C. Moore, M. John, I. James, F. Christiansen, C. Witt, and S. Mallal. Evidence of HIV-1 adaptation to HLA-restricted immune responses at a population level. *Science*, 296:1439–1443, 2002.

[9]  R. Neal and G. Hinton. A view of the EM algorithm that justifies incremental, sparse, and other variants. In *Learning in graphical models*, M. Jordan ed. (MIT Press,1999).

[10]  H Rammensee, J Bachmann, N P Emmerich, O A Bachor, and S Stevanovic. SYFPEITHI: database for MHC ligands and peptide motifs. *Immunogenetics*, 50(3-4):213–219, Nov 1999.
